# Characterizing response behavior in multi-sensory perception with conflicting cues

**Rama Natarajan**[1]     **Iain Murray**[1]     **Ladan Shams**[2]     **Richard S. Zemel**[1]
[1]Department of Computer Science, University of Toronto, Canada
`{rama,murray,zemel}@cs.toronto.edu`
[2]Department of Psychology, University of California Los Angeles, USA
`ladan@psych.ucla.edu`

## Abstract

We explore a recently proposed mixture model approach to understanding interactions between conflicting sensory cues. Alternative model formulations, differing in their sensory noise models and inference methods, are compared based on their fit to experimental data. Heavy-tailed sensory likelihoods yield a better description of the subjects' response behavior than standard Gaussian noise models. We study the underlying cause for this result, and then present several testable predictions of these models.

## 1  Introduction

A natural scene contains several multi-modal sensory cues to the true underlying values of its physical properties. There is substantial evidence that the brain deals with the sensory information from multiple modalities simultaneously, to form a coherent and unified percept of the world and to guide action. A major focus of multi-sensory perceptual studies has been in exploring the synergistic as well as modulatory interactions between individual sensory cues. The perceptual consequences of these interactions can be effectively explored in cases where the cues are in conflict with each other, resulting in potentially illusory percepts such as the "ventriloquism effect" [1].

A well-tested hypothesis with regards to multi-sensory cue interaction is that the individual sensory estimates are combined in a linear fashion, weighted by their relative reliabilities. Most studies that expound this linear approach assume that sensory noise in the different modalities are independent of each other, and that the sensory likelihoods can be well approximated by Gaussian distributions. Under these assumptions, the maximum-likelihood estimator of the underlying physical variable is an affine combination of the sensory estimates weighted in proportion to their precisions. This linear model predicts that the variance of the posterior distribution is always lower than that of individual cues. However, data from several psychophysical studies contradict this prediction, necessitating non-linear computational strategies to deal with the inputs.

Recent studies [2; 3; 4; 5] have proposed a particular form of mixture model to address response behavior in situations with a large conflict between sensory stimuli. Conflicts arise when corresponding cues suggest very different estimates of an underlying variable. The basic intuition behind these models is that large stimulus disparities might be a consequence of the stimuli having resulted from multiple underlying causal factors. We evaluate the different formulations in their ability to model experimental data [6] that exhibit very interesting non-linear response behavior under conflicting stimulus conditions. The formulations differ in how perceptual estimates are derived from sensory data. We demonstrate some inadequacies of the current models and propose an alternative formulation that employs heavy-tailed sensory likelihoods. The proposed model not only achieves better fits to non-linear response behavior in the experimental data but also makes several quantitatively testable predictions.

## 2 A Mixture Model for Evaluating Cue Interactions

In this section, we present an overview of a recently proposed mixture model approach [3] to dealing with conflicting sensory inputs. We describe two approaches to inference under this model — causal averaging and causal selection — and analyze the model predictions on our simulation of an auditory localization task [6].

The environmental variables of interest are the spatial locations of an auditory and visual stimulus, denoted by $s_a$ and $s_v$ respectively. Information about the stimuli is provided by noisy sensory cues $x_a$ and $x_v$. The model evaluates sensory cues under two discrete hypotheses ($C = \{1, 2\}$) regarding the causal structure underlying the generation of the stimuli. The hypotheses are that the two stimuli could arise from the same ($C = 1$) or different ($C = 2$) causal events. This mixture model instantiates a simple idea: if there is a common cause, cues are combined; otherwise they are segregated. The model is characterized by (i) the sensory likelihoods $P(x_v|s_v)$ and $\mathrm{P}(x_a|s_a)$, (ii) the prior distributions $P(s_v, s_a)$ over true stimulus positions and (iii) the prior over hypotheses $P(C)$.

### 2.1 Generating sensory data

The standard model assumes Gaussian sensory likelihoods and prior distributions. The true auditory and visual stimulus positions are assumed to be the same for $C = 1$, i.e., $s_a = s_v = s$ drawn from a zero-mean Gaussian prior distribution: $s \sim \mathcal{N}(0, \sigma_p^2)$ where $\sigma_p$ is standard deviation of the distribution. The noisy sensory evidence $x_a$ is a sample from a Gaussian distribution with mean $s_a = s$ and standard deviation $\sigma_a$: $x_a \sim \mathcal{N}(x_a; s_a = s, \sigma_a^2)$. Similarly for the visual evidence: $x_v \sim \mathcal{N}(x_v; s_v = s, \sigma_v^2)$.

When there are $C = 2$ underlying causes, they are drawn independently from the zero-mean Gaussian prior distribution: $s_v \sim \mathcal{N}(0, \sigma_p^2)$; $s_a \sim \mathcal{N}(0, \sigma_p^2)$. Then $x_v \sim \mathcal{N}(x_v; s_v, \sigma_v^2)$ and $x_a \sim \mathcal{N}(x_a; s_a, \sigma_a^2)$. The belief in each hypothesis given the cues $x_a$ and $x_v$ is defined by the posterior distribution:

$$P(C|x_v, x_a) \quad = \quad \frac{P(x_v, x_a|C)P(C)}{P(x_v, x_a)} \tag{1}$$

When the hypotheses are discrete $C = \{1, 2\}$, the normalization constant $P(x_v, x_a) = P(x_v, x_a|C = 1)P(C = 1) + P(x_v, x_a|C = 2)(1 - P(C = 1))$.

Given this particular causal generative model, the conditional likelihoods in Equation 1 are defined as $P(x_v, x_a|C = 1) = \int P(x_v|s_v = s)P(x_a|s_a = s)P(s)ds$ and $P(x_v, x_a|C = 2) = \int P(x_v|s_v)P(s_v)ds_v \int P(x_a|s_a)P(s_a)ds_a$. The conditional sensory likelihoods are specified as: $P(x_v, x_a|s_v, s_a, C) = P(x_v|s_v)P(x_a|s_a)$.

### 2.2 Inference methods

#### 2.2.1 Causal averaging

The conditional posterior over stimulus variables is calculated for each hypothesis as $P(s_v, s_a|x_v, x_a, C = 1)$ and $P(s_v, s_a|x_v, x_a, C = 2)$. The standard approach to computing the full posterior distribution of interest $P(s_a, s_v|x_a, x_v)$ is by integrating the evidence over both hypotheses weighted by the posterior distribution over $C$ (Equation 1). Such a *model averaging* approach to causal inference is specified by the following identity:

$$P_{\mathrm{avg}}(s_v, s_a|x_v, x_a) \quad = \quad \sum_C P(s_v, s_a|x_v, x_a, C)P(C|x_v, x_a) \tag{2}$$

$$= \quad \sum_C \frac{P(x_v, x_a|s_v, s_a, C)P(s_v, s_a|C)P(C|x_v, x_a)}{P(x_v, x_a|C)} \tag{3}$$

Here, $P(C = 1|x_v, x_a) = \pi_c$ is the posterior mixing proportion and $(1 - \pi_c) = P(C = 2|x_v, x_a)$.

### 2.2.2 Causal selection

An alternative approach is to calculate an approximate posterior distribution by first selecting the hypothesis $C^*$ that maximizes the posterior distribution $P(C|x_v, x_a)$. Under this *model selection* approach, subsequent inference is based on the selected hypothesis alone.

$$C^* = \underset{C=\{1,2\}}{\operatorname{argmax}} P(C|x_v, x_a) \tag{4}$$

Then the posterior distribution over stimulus location is approximated as follows:

$$
\begin{aligned}
P_{\text{sel}}(s_v, s_a|x_v, x_a) &\approx P(s_v, s_a|x_v, x_a, C = C^*) \tag{5}\\
&= \frac{P(x_v, x_a|s_v, s_a, C = C^*)P(s_v, s_a|C = C^*)}{P(x_v, x_a|C = C^*)} \tag{6}
\end{aligned}
$$

## 2.3 Evaluating the models on experimental data

Here, we evaluate the causal averaging and selection models on an auditory localization task [6] where visual and auditory stimuli were presented at varying spatial and temporal disparities. In addition to reporting the location of the auditory target, subjects were also asked to report on whether they perceived the two stimuli to be perceptually unified. The variables examined were the bias and variance of the subjects' estimates for each stimulus condition. The data exhibit very interesting non-linear response behavior (solid lines in Figures 1A and 1D).

In our simulation of the task, the auditory target was presented at locations $\{0°, 5°, 10°\}$ left or right of fixation. Although the real experiment varied the fixation location from trial to trial, it was found to have no effect on subsequent analyses and data were collapsed across all fixation locations. Hence, we assume the fixation point to be at the center of space $(0°)$. The visual stimuli were assumed to be temporally coincident with the auditory stimuli and presented at varying spatial disparities $\{0°, 5°, 10°, 15°, 20°, 25°\}$ left or right of sound. Sensory evidence $x_a$ and $x_v$ were corrupted by Gaussian noise as described earlier.

Each stimulus combination $\{s_a, s_v\}$ was presented with equal probability 2000 times. The spatial axis ranged from $-25°$ to $25°$ and was divided into $1°$ width bins. On each trial, the model computes a posterior probability distribution over stimulus locations conditioned on the noisy cues $x_a$ and $x_v$ according to one of Equations 3 or 6. It then estimates visual and auditory locations $\hat{s}_a$ and $\hat{s}_v$ as the peak of the posterior distribution (maximum *aposteriori* estimate): $\hat{s}_a = \operatorname{argmax}_{s_a} P(s_a, s_v|x_a, x_v)$.

We have simulated estimators using other criteria, such as minimizing the squared error of the estimates (i.e, expected value of the posterior distribution). The results were very similar using the different estimators. Percent bias is given by: $\frac{\hat{s}_a - s_a}{s_v - s_a} * 100$. Goodness of fit was computed using squared error loss to quantify the amount by which model estimates differed from the behavioral data. For analysis, the trials were dichotomized into unity and non-unity trials based on the perception of spatial unity. A trial was classified as *unity* if the posterior probability $P(C = 1|x_v, x_a)$ was greater than some threshold $\rho$ and *non-unity* otherwise.

The simulation results (i.e., the estimates $\hat{s}_a$ and $\hat{s}_v$) were averaged across trials in each category. The parameters of the model are: 1) the stimulus location variance $\sigma_p^2$, 2–3) the observation variances $\sigma_a^2$ and $\sigma_v^2$, 4) the prior mixture proportion $\omega = P(C=1)$, and 5) the unity perception threshold $\rho$. The parameter values were estimated to fit the experimental data and are provided in the figure captions.

## 2.4 Simulation results for the Gaussian model

Figure 1 presents predictions made by both the theoretical models. The behavioral data [6] (solid lines in all plots) range from spatial disparities $-15°$ to $15°$; error bars represent standard errors across 5 subjects. Model predictions (dashed lines) extend to a wider range of $-25°$ to $25°$. Some of the predicted trends are similar to the behavioral data. Regardless of stimulus disparity, whenever visual and auditory stimuli were perceived as unity,

the predicted response bias was very high (dashed gray; Figure 1A). This means that the auditory location was perceived to be very near to the visual stimulus. When the stimuli appeared to not be unified, the auditory location was biased away from the visual stimulus — increasingly so as disparity decreased (dashed black; Figure 1A).

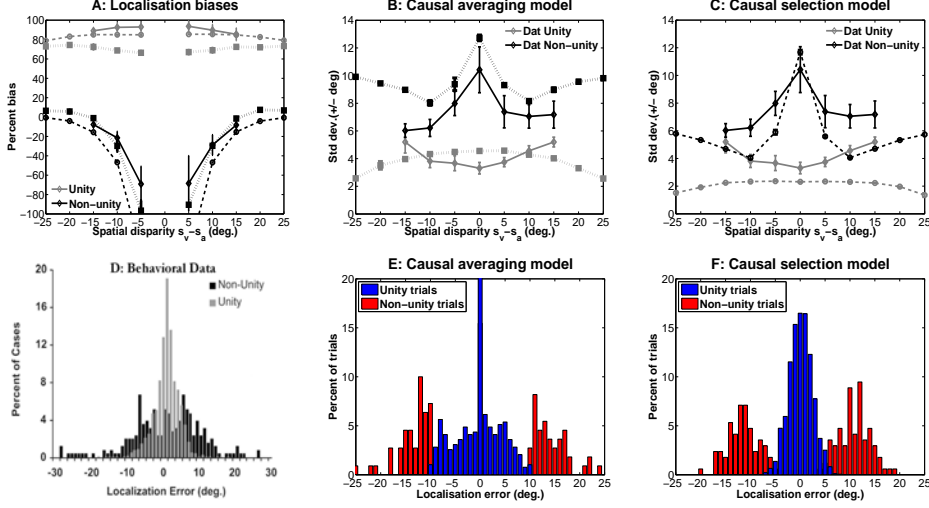

Figure 1: **Simulation results - Gaussian sensory likelihoods:** In this, and all subsequent figures, solid lines plot the actual behavioral data reported in [6] and dashed lines are the model predictions. (A) Localization biases in the data, plotted alongside predictions from both models. (B) Causal averaging model, response variability: $\sigma_a = 8$, $\sigma_v = 0.05$, $\omega = 0.15$. (C) Causal selection model: $\sigma_a = 6$, $\sigma_v = 2.5$, $\omega = 0.2$. For both models: $\sigma_p = 100$, $\rho = 0.5$. (D) Distribution of localization errors in data, for $s_v - s_a = 0$; re-printed with permission from [6]. (E,F) Localization errors predicted by the causal averaging and causal selection models respectively.

However, both the models exhibit one or more significant differences from the experimental observations. The predicted curves for unity trials (dashed gray; Figures 1B,C) are all concave, whereas they were actually observed to be convex (solid gray lines). On non-unity trials too, the predicted response variabilities (dashed black lines) are an inadequate fit to the real data (solid black lines).

An additional test for the appropriateness of the models is the predictions they make with regards to the distribution of localisation errors. An analysis of the behavioral data derived from the spatially coincident stimulus conditions ($s_v - s_a = 0$) revealed a distinct pattern (Figure 1D). On unity trials, localization error was $0°$ implying that the responses were clustered around the auditory target. On non-unity trials, the errors were bi-modally distributed and failed the test for normality [6]. Causal selection predicts a qualitatively similar distribution of errors (Figure 1F), suggesting that it may be the most appropriate inference strategy under the given task and model assumptions.

## 3   An Alternative Model for Sensory Likelihoods

### 3.1   Heavy-tailed likelihood formulation

In this section, we re-formulate the sensory likelihoods $P(x_a|s_a)$ and $P(x_v|s_v)$ as a mixture of Gaussian and uniform distributions. This mixture creates a likelihood function with heavy tails.

$$x_v \sim \pi\mathcal{N}(x_v; s_v, \sigma_v^2) + \frac{(1-\pi)}{r_l}; x_a \sim \pi\mathcal{N}(x_a; s_a, \sigma_a^2) + \frac{(1-\pi)}{r_l} \qquad (7)$$

### 3.2   Simulation results with heavy-tailed sensory likelihoods

Figure 2 presents predictions made by the theoretical models based on heavy-tailed likelihoods. Both models now provide a much better fit to bias and variance, compared to their

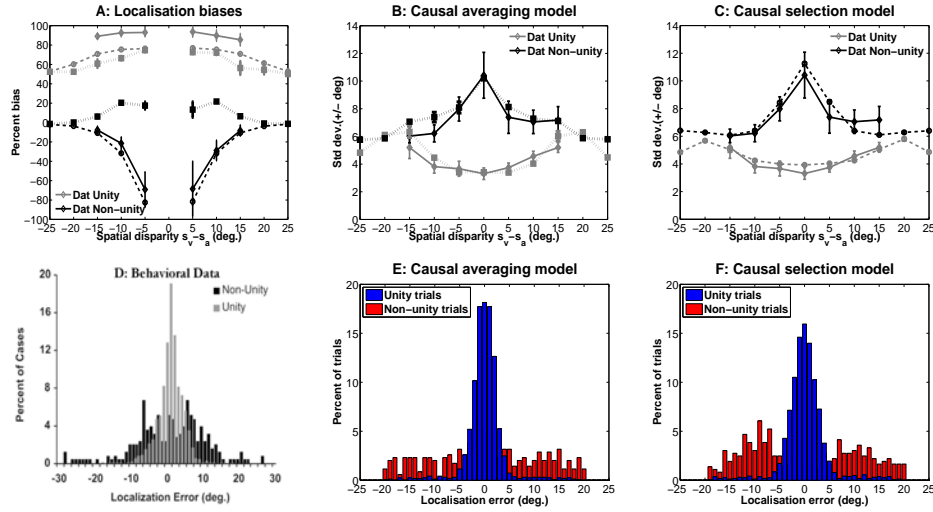

Figure 2: **Simulation results - heavy-tailed likelihoods:** (A) Localization biases in the data, plotted alongside model predictions. (B) Causal averaging model, response variability: $\sigma_a = 3.5$, $\sigma_v = 2$. (C) Causal selection model: $\sigma_a = 5$, $\sigma_v = 2.5$. In both models, $\sigma_p = 100$, $\omega = 0.2$, $\rho = 0.5$, $r_l = 180°$. (D) Distribution of localization errors in data, for $s_v - s_a = 0$. (E,F) Localization errors predicted by the heavy-tailed causal averaging and causal selection models.

Gaussian counterparts. The heavy-tailed causal averaging model (Figure 2B) makes reasonable predictions with regards to variability. However, both the amount and the trend of predicted biases for non-unity trials (dotted line; 2A) do not match observations.

Here too, the best-fitting model is causal selection (dashed line; Figures 2A,C). The localization error distribution (Figure 2F) very closely matches the true observations (Figure 2D) in how the unity responses are uni-modally distributed about the target location $s_a$, and non-unity responses are bi-modally distributed either side of the target. Visually, this is a better prediction of the true distribution of errors, compared to the prediction made by the Gaussian causal selection model (Figure 1F); we are unable to make a quantitative comparison for want of access to the raw data.

Compared with the results in Figure 1, our models make very different bias and variance predictions for spatial disparities not tested. This is discussed in detail in Section 4. The heavy-tailed likelihood model has two more free parameters ($r_p$ and mixing proportion $\pi$; Equation 7) than the Gaussian, which is essentially a subset of the heavy-tailed mixture when $\pi = 1$. Although the Gaussian model may be preferred for its computational simplicity, it is a demonstrably poor fit to the data and the heavy-tailed model is a worthwhile improvement.

### 3.3 Analyzing the likelihood models

Existence of the heavy tails in the likelihood function seems to be a critical feature that supports the non-linear behavior in the data. We substantiate this suggestion using Figure 3, and attempt to give some intuition behind the qualitative differences in variability and bias between Figures 1 and 2. The discussion below focuses on 3 disparity conditions. The congruent case $|s_v - s_a| = 0$ is chosen for reference; $|s_v - s_a| = 10$ and $|s_v - s_a| = 25$ are chosen since the Gaussian and heavy-tailed models tend to differ most in their predictions at these disparities.

Let us first consider the unity case. In general, most of the samples on unity trials are from the region of space where both the auditory and visual likelihoods overlap. When true disparity $|s_v - s_a| = 0$, it means that the two likelihoods overlap maximally (Figures 3Aii and 3Cii). Hence regardless of the form of the likelihood, variability on unity trials at $|s_v - s_a| = 0$ should be roughly between $\sigma_v$ and $\sigma_a$. This can be verified in Figures 1C, 2C.

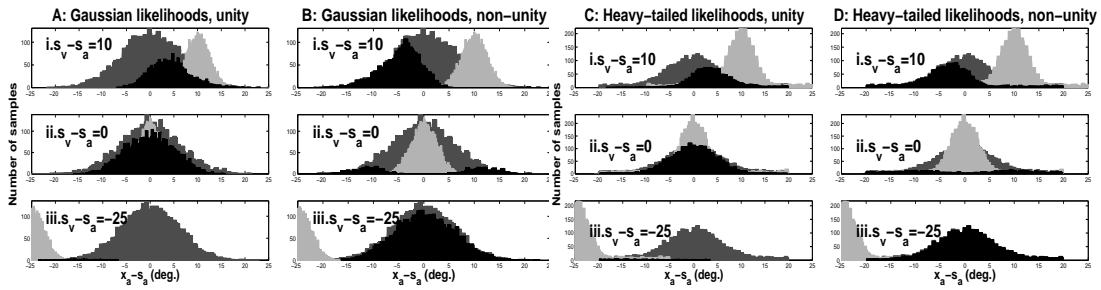

Figure 3: **Analyzing the likelihood models:** Results from the causal selection models. In all plots, light-gray histograms are samples $x_v$ from visual likelihood distribution; dark-gay histograms plot $x_a$. Black histograms are built only from samples $x_a$ on which either unity (A,C) or non-unity (B,D) judgment was made. Each panel corresponds to one of three chosen disparities; histograms in the panel plot samples from all stimulus conditions that correspond to that particular disparity.

Now one of the biggest differences between the likelihood models is what happens to this variability as $|s_v - s_a|$ increases. In the case of the Gaussian, the amount of overlap between the two likelihoods decreases (Figures 3Ai,3Aiii). Consequently, the samples are from a somewhat smaller region in space and hence the variability also decreases. This corresponds to the concave curves predicted by the Gaussian model (Figures 1C; dashed gray). Whereas for the heavy-tailed likelihood, the overlapping regions roughly increase with increasing disparity, due to the long tails (Figures 3Ci,3Ciii). This is reflected in the gradually increasing variability on unity trials corresponding to the better matching convex curves predicted by the heavy-tailed model (Figure 2C).

On the non-unity trials, most of the samples are from non-overlapping regions of space. Here, the biggest difference between the likelihood models is that in the Gaussian case, after a certain spatial limit, the variability tends to increase with increasing $|s_v - s_a|$. We also see this trend in simulation results presented in [2; 4]. This is because as disparity increases, the degree of overlap between two likelihoods decreases and variability approaches $\sigma_a$ (Figures 3Bi,3Biii). However, the behavior in the real data suggests that variability continues to be a constant. With heavy-tailed likelihoods, the tails of the two likelihoods continue to overlap even as disparity increases; hence the variability is roughly constant (Figures 3Di,3Diii).

## 4 Model Predictions

**Quantitative predictions—variance and bias:** Our heavy-tailed causal selection model makes two predictions with regards to variability and bias for stimulus conditions not yet tested. One prediction is that on non-unity trials, as spatial disparity $s_v - s_a$ increases, the localisation variability continues to remain constant at roughly a value equivalent to the standard deviation of the auditory likelihood (Figure 2C; black dashed plot). However, response percent bias approaches zero (Figure 2A; black dashed plot), indicating that when spatial disparity is very high and the stimuli are perceived as being independent, auditory localisation response is consistent with auditory dominance.

A second prediction is that percent bias gradually decreases with increasing disparity on unity trials as well. This suggests that even when highly disparate stimuli are perceived as being unified, perception may be dominated by the auditory cues. Our results also predict that the variability in this case continues to increase very gradually with increasing disparity up to some spatial limits ($|s_v - s_a| = 20°$ in our simulations) after which it begins to decrease. This accords with intuition, since for very large disparities, the number of trials in which the the stimuli are perceived as being unified will be very small.

**Qualitative prediction—distribution of localization errors:** Our model also makes a qualitative prediction concerning the distribution of localisation errors for incongruent ($s_v - s_a \neq 0$) stimulus conditions. In both Figures 4A and B, localization error on unity trials is equivalent to the stimulu disparity $s_v - s_a = 10°$, indicating that even at this high disparity, responses are cluttered closer to the visual stimulus location. On non-unity trials, the error

is about $5°$ here; responses are more broadly distributed and the bias is highly reduced. The Gaussian and heavy-tailed predictions differ in how quickly the error distributions go to zero.

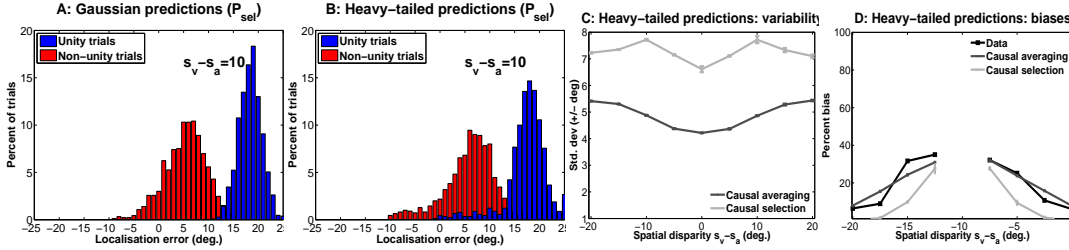

Figure 4: **Model predictions:** (A,B) Localization error distributions predited by the Gaussian and heavy-tailed causal selection models. Plots correspond to stimulus condition $s_v = 20; s_a = 10$. (C,D) Response variability and bias predicted by they heavy-tailed causal averaging and selection models on simulation of an audio-visual localization task [3].

**Specificity to experimental task:** In the experimental task we have examined here [6], subjects were subjects were asked to first indicate the perceived location of sound on each trial and then to report their judgement of unity. The requirement to explicitly make a unity judgement may incur an experimental bias towards the causal selection model.

To explore the potential influence of task instructions on subjects' inference strategy, we tested our models on a simulation of a different audio-visual spatial localisation task [3]. Here, subjects were asked to report on both visual and auditory stimulus locations and were not explicitly instructed to make unity judgements. The authors employed model averaging to explain the results [3] and the data were found to have a very high likelihood under their model. However, they do not analyse variability in the subjects' responses and this aspect of behavior as a function of spatial disparity is not readily obvious in their published data.

We evaluated both our heavy-tailed causal averaging as well as causal selection models on a simulation of this experiment. The two models make very different predictions. Causal averaging predicts that response variability will monotonically increase with increasing disparity, while selection predicts a less straightforward trend (Figure 4C). Both models predict a similar amount of response bias and that it will decrease with increasing disparity (Figure 4C). This particular prediction is confirmed by the response bias in their behavioral data plot made available in [3]. Considering the paradigmatic differences between the two studies ([6] and [3]) and the wide range in bias, applying both inference methods and likelihood models on this data could be very informative.

**Adaptation of the prior:** One interesting aspect of inference under this generative model is that as the value of $\omega = P(C = 1)$ increases, the variability also increases for both unity and non-unity trials across all disparities. However, the response bias remains unchanged. Given this correlation between response variability and the prior over hypotheses, our approach may be used to understand whether and how subjects' priors change during the course of an experimental session. Considering that the best value across all trials for this prior is quite small ($\omega \sim 0.2$), we hypothesize that this value will be quite high at the start of an experiment, and gradually reduce. This hypothesis leads to a prediction that variability decreases during an experimental session.

## 5 Discussion

In this paper, we ventured to understand the computational mechanisms underlying sensory cue interactions that give rise to a particular pattern of non-linear response behavior [6], using a mixture of two different models that could have generated the sensory data. We proposed that the form of the sensory likelihood is a critical feature that drives non-linear behavior, especially at large stimulus disparities. In particular, a heavy-tailed likelihood function more accurately fits subjects' bias and variance in a cue combination task.

Heavy-tailed distributions have been used previously in modeling cue interactions [7; 8]. In this paper, we went further by comparing the ability of heavy-tailed and Gaussian like-

lihood models to describe behavior. Qualitative fits of summarised statistics such as bias and variance are insufficient to make any strong claims about human perceptual processes; nevertheless, this work provides some insight into the potential functional role of sensory noise.

Another significant contribution in this paper is the critical evaluation of model selection versus averaging approaches to inference. These two inference methods may predict different variances in their estimates, as a function of stimulus conflict. As suggested in Section 4, having these different models at hand allows one to examine how task instructions affect subject behavior.

We noted in Section 3.2 that the heavy-tailed model is more complex than the Gaussian model. Although we have not included any complexity penalty, this formulation was supported by two aspects: (i) it was relatively insensitive to parameter settings, providing a better fit to the data than the Gaussian model for a wide range of parameter values; (ii) optimizing the fit of the Gaussian model required implausible values for parameters $\sigma_a$, $\sigma_v$ (Fig 1B), whereas parameters for the heavy-tailed model accorded well with published data.

One downside about our results is that even though the model bias for unity trials captures the slightly increasing trend as disparity decreases, it is not as large as in the behavioral data (close to $100\%$) or as that predicted by the Gaussian models. This does not seem to be a consequence of the parameter values chosen. One interpretation provided by [6] of the large bias in the data is that a perceptual decision (unity or non-unity) determines a sensorimotor action (localization response). Then one response strategy might be to ignore the posterior probability $P(s_a|x_v, x_a)$ once unity is judged and then set $\hat{s}_a = \hat{s}_v$; although this results in prediction of higher bias, the strategy is not Bayes-optimal. Yet another potential limitation of our approach is that the only form of noise we consider is sensory; we do not yet take into account any motor component that may drive target localization.

Currently, we have access to only an estimate of the average variance in subjects' auditory target location estimates. On the computational side, one interesting avenue for future work would be to evaluate the model averaging and selection hypothesis based on a likelihood model derived directly from the raw data. On the experimental side, one of the major inadequacies of most experimental paradigms is that the only (approximate) measure of a subject's perceptual uncertainty involves measuring the response variability across a large number of trials. An alternative paradigm that allows measurement of the perceptual uncertainty on a single trial could provide important constraints on computational models of the perceptual phenomena. At the neural level, a key step entails exploring biologically plausible neural implementations of the mixture model approach.

### Acknowledgments

The authors would like to thank National Sciences and Engineering Research Council of Canada and Canadian Institute For Advanced Research (RN and RZ), the government of Canada (IM), UCLA Faculty Grants Program and UCLA Faculty Career Development (LS).

### References

[1] I P Howard and W B Templeton. *Human spatial orientation*. Wiley, New York, 1966.

[2] Konrad P Körding and Joshua B Tenenbaum. Causal inference in sensorimotor integration. In *NIPS*, pages 737–744. MIT Press, 2006.

[3] Konrad P Körding, Ulrik Beierholm, Wei Ji Ma, Steven Quartz, Joshua B Tenenbaum, and Ladan Shams. Causal inference in multisensory perception. *PLoS ONE,* 2(9), 2007.

[4] Y Sato, T Toyoizumi, and K Aihara. Bayesian inference explains perception of unity and ventriloquism aftereffect. *Neural Comp.,* 19:3335–55, 2007.

[5] Alan Stocker and Eero Simoncelli. A Bayesian model of conditioned perception. In *NIPS 20*, pages 1409–1416. MIT Press, Cambridge, MA, 2008.

[6] MT Wallace, GE Roberson, WE Hairston, BE Stein, JW Vaughan, and JA Schirillo. Unifying multisensory signals across time and space. *Exp Brain Res.,* 158(2):252–8, 2004.

[7] David C Knill. Robust cue integration: A Bayesian model and evidence from cue-conflict studies with stereoscopic and figure cues to slant. *Journal of Vision,* 7(7):1–24, 2007.

[8] Alan A Stocker and Eero P Simoncelli. Noise characteristics and prior expectations in human visual speed perception. *Nat. Neurosci.,* 9:578–585, 2006.
